# A systematic approach to extracting semantic information from functional MRI data

**Francisco Pereira**
Siemens Corporation, Corporate Technology
Princeton, NJ 08540
francisco.pereira@gmail.com

**Matthew Botvinick**
Princeton Neuroscience Institute and Department of Psychology
Princeton University
Princeton NJ 08540
matthewb@princeton.edu

## Abstract

This paper introduces a novel classification method for functional magnetic resonance imaging datasets with tens of classes. The method is designed to make predictions using information from as many brain locations as possible, instead of resorting to feature selection, and does this by decomposing the pattern of brain activation into differently informative sub-regions. We provide results over a complex semantic processing dataset that show that the method is competitive with state-of-the-art feature selection and also suggest how the method may be used to perform group or exploratory analyses of complex class structure.

## 1 Introduction

Functional Magnetic Resonance Imaging (fMRI) is a technique used in psychological experiments to measure the blood oxygenation level throughout the brain, which is a proxy for neural activity; this measurement is called *brain activation*. The data resulting from such an experiment is a 3D grid of cells named *voxels* covering the brain (on the order of tens of thousands, usually), measured over time as tasks are performed and thus yielding one time series per voxel (collected every 1-2 seconds and yielding hundreds to thousands of points).

In a typical experiment, brain activation is measured during a task of interest, e.g. reading words, and during a related control condition, e.g. reading nonsense words, with the goal of identifying brain locations where the two differ. The most common analysis technique for doing this – statistical parametric mapping [4] – tests each voxel individually by regressing its time series on a predicted time series determined by the task contrast of interest. This fit is scored and thresholded at a given statistical significance level to yield a brain image with clusters of voxels that respond very differently to the two tasks (colloquially, these are the images that show parts of the brain that "light up"). Note, however, that for both tasks there are many other processes taking place in tandem with this task-contrasting activation: visual processing to read the words, attentional processing due to task demands, etc. The output of this process for a given experiment is a set of 3D coordinates of all the voxel clusters that appear reliably across all the subjects in a study. This result is easy to interpret, since there is a lot of information about what processes each brain area may be involved in. The coordinates are comparable across studies, and thus result reproduciblity is also an expectation.

In recent years, there has been increasing awareness of the fact that there is information in the entire pattern of brain activation and not just in saliently active locations. Classifiers have been the tool

of choice for capturing this information and used to make predictions ranging from what stimulus a subject is seeing, what kind of object they are thinking about or what decision they will make [12] [14] [8]. The most common situation is to have an example correspond to the average brain image during one or a few performances of the task of interest, and voxels as the features, and we will discuss various issues with this scenario in mind.

The goal of this work is generally not (just) classification accuracy per se, even in diagnostic applications, but understanding where the information used to classify is present. If only two conditions are being contrasted this is relatively straightforward as information is, at its simplest, a difference in activation of a voxel in the two conditions. It's thus possible to look at the magnitudes of the weights a classifier puts on voxels across the brain and thus locate the voxels with the largest weights [1]; given that there are typically two to three orders of magnitude more voxels than examples, though, classifiers are usually trained on a selection of voxels rather than the entire activation pattern. Often, this means the best accuracy is obtained using few voxels, from all across the brain, and that different voxels will be chosen in different cross-validation folds; this presents a problem for interpretability of the locations in question.

One approach to this problem is to try and regularize classifiers so that they include as many informative voxels as possible [2], thus identifying localizable clusters of voxels that may overlap across folds. A different approach is to cross-validate classifiers over small sections of the grid covering the brain, known as *searchlights* [10]. This can be used to produce a map of the cross-validated accuracy in the searchlight around each voxel, taking advantage of the pattern of activation across all the voxels contained in it. Such a map can then be thresholded to leave only locations where accuracy is significantly above chance. While these approaches have been used successfully many times over the last decade, they will become progressively less useful in face of the increasing commonality of datasets with tens to hundreds of stimuli, and a correspondingly high number of experimental conditions. Knowing the location of a voxel does not suffice to interpret what it is doing, as it could be very different from stimulus to stimulus (rather than just active or not, as in the two condition situation). It's also likely that no small brain regions will allow for a searchlight classifier capable of distinguishing between all possible conditions at the spatial resolution of fMRI, and hence defining a searchlight size or shape is a trade-off between including voxels and making it harder to locate information or train a classifier – as the number of features increases as the number of examples remains constant – and excluding voxels and thus the number of distinctions that can be made.

This paper introduces a method to address all of these issues while still yielding an interpretable, whole-brain classifier. The method starts by learning how to decompose the pattern of activation across the brain into sub-patterns of activation, then it learns a whole-brain classifier in terms of the presence and absence of certain subpatterns and finally combines the classifier and pattern information to generate brain maps indicating which voxels belong to informative patterns and what kind of information they contain. This method is partially based on the notion of pattern feature introduced in an earlier paper by us [15], but has been developed much further so as to dispense with most parameters and allow the creation of spatial maps usable for group or exploratory analyses, as will be discussed later.

## 2 Data and Methods

### 2.1 Data

The grid covering the brain contains on the order of tens of thousands voxels, measured over time as tasks are performed, every 1-2 seconds, yielding hundreds to thousands of 3D images per experiment. During an experiment a given task is performed a certain number of times – trials – and often the images collected during one trial are collapsed or averaged together, giving us one 3D image that can be clearly labeled with what happened in that trial, e.g. what stimulus was being seen or what decision a subject made. Although the grid covers the entire head, only a fraction of its voxels contain cortex in a typical subject; hence we only consider these voxels as features.

A *searchlight* is a small section of the 3D grid, in our case a $27 = 3 \times 3 \times 3$ voxel cube. Analyses using searchlights generally entail computing a statistic [10] or cross-validating a classifier over the dataset containing just those voxels [16], and do so for the searchlight around each voxel in the brain, covering it in its entirety. The intuition for this is that individual voxels are very noisy features, and an effect observed across a group of voxels is more trustworthy.

In the experiment performed to obtain our dataset [2] [13], subjects observed a word and a line drawing of an item, displayed on a screen for 3 seconds and followed by 8 seconds of a blank screen. The items named/depicted belonged to one of 12 categories: animals, body parts, buildings, building parts, clothing, furniture, insects, kitchen, man-made objects, tools, vegetables and vehicles. The experimental task was to think about the item and its properties while it was displayed. There were 5 different exemplars of each of the 12 categories and 6 experimental epochs. In each epoch all 60 exemplars were shown in random order without repetition, and all epochs had the same exemplars. During an experiment the task repeated a total of 360 times, and a 3D image of the fMRI-measured brain activation acquired every second.

Each example for classification purposes is the average image during a 4 second span while the subject was thinking about the item shown a few seconds earlier (a period which contains the peak of the signal during the trial; the dataset thus contains 360 examples, as many as there were trials. The voxel size was $3 \times 3 \times 5$ mm, with the number of voxels being between 20000 and 21000 depending on which of the 9 subjects was considered. The features in each example are voxels, and the example labels are the category of the item being shown in the trial each example came from.

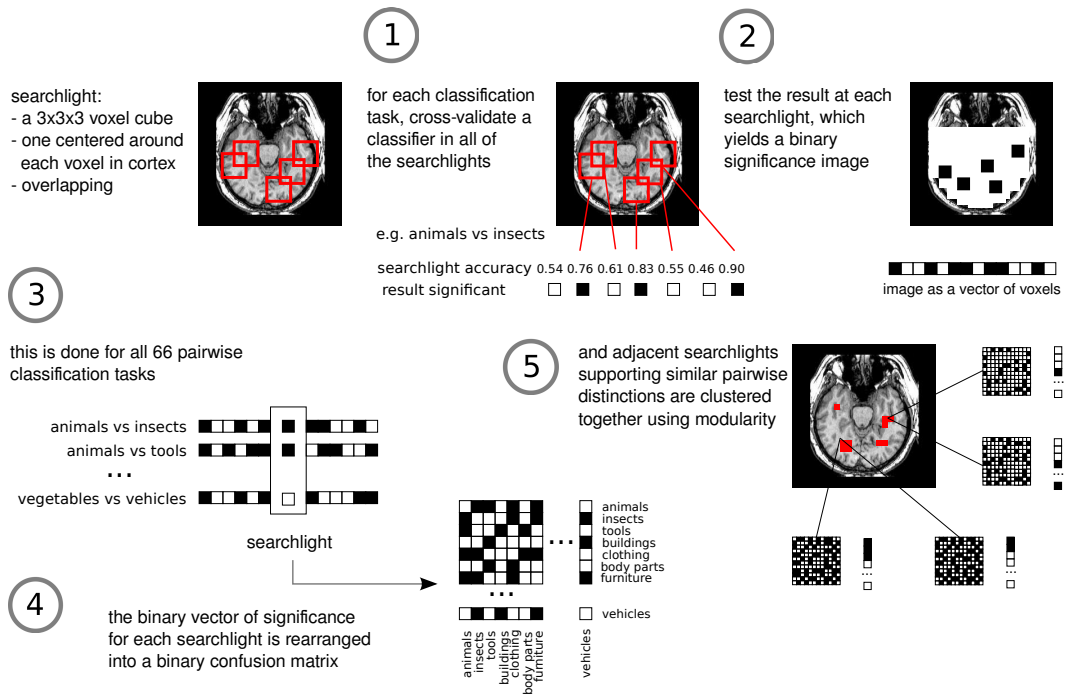

Figure 1: Construction of data-driven searchlights.

## 2.2 Method

The goal of the experiment our dataset comes from is to understand how a certain semantic category is represented throughout the brain (e.g. do "Insects" and "Animals" share part of their representation because both kinds of things are alive?). Intuitively, there is information in a given location if at least two categories can be distinguished looking at their respective patterns of activation there; otherwise, the pattern of activation is noise or common to all categories. Our method is based upon this intuition, and comprises three stages:

1. the construction of data-driven searchlights, parcels of the 3D grid where *the same discriminations between pairs of categories can be made* (these are generally larger than the $3 \times 3 \times 3$ basic searchlight)

2. the synthesis of *pattern features* from each data-driven searchlight, corresponding to the presence or absence of a certain pattern of activation across it

3. the training and use of a classifier based on pattern features and the generation of an anatomical map of the impact of each voxel on classification

and these are described in detail in each of the following sections.

### 2.2.1 Construction of data-driven searchlights

**Create pairwise searchlight maps**   In order to identify informative locations we start by considering whether a given pair of categories can be distinguished in each of the thousands of $3 \times 3 \times 3$ searchlights covering the brain:

1. For each searchlight cross-validate a classifier using the voxels belonging to it, obtaining an accuracy value which will be assigned to the voxel at the center of the searchlight, as shown in part 1 of Figure 1. The classifier used in this case was Linear Discriminant Analysis (LDA, [7]), with a shrinkage estimator for the covariance matrix [18], as this was shown to be effective at both modeling the joint activation of voxels in a searchlight and classification [16].

2. Transform the resulting brain image with the accuracy of each voxel into a $p$-value brain image (of obtaining accuracy as high or higher under the null hypothesis that the classes are not distinguishable, see [11]), as shown in part 1 of Figure 1.

3. Threshold the $p$-value brain image using False Discovery Rate [5] ($q = 0.01$) to correct multiple for multiple comparisons and get a binary brain image with candidate locations where this pair of categories can be distinguished, as shown in part 2 of Figure 1.

The outcome for each pair of categories is a binary significance image, where a voxel is 1 if the categories can be distinguished in the searchlight surrounding it or 0 if not; this is shown for all pairs of categories in part 3 of Figure 1. This can also be viewed per-searchlight, yielding a binary vector encoding which category pairs can be distinguished and which can be rearranged into a binary matrix, as shown in part 4 of Figure 1.

**Aggregate adjacent searchlights**   Examining each small searchlight makes sense if we consider that, a priori, we don't know where the information is or how big a pattern of activation would have to be considered (with some exceptions, notably areas that respond to faces, houses or body parts, see [9] for a review). That said, if the same categories are distinguishable in two adjacent searchlights – which overlap – then it is reasonable to assume that all their voxels put together would still be able to make the same distinctions. Doing this repeatedly allows us to find *data-driven* searchlights, not bound by shape or size assumptions. At the same time we would like to constrain data-driven searchlights to the boundaries of known, large, anatomically determined regions of interest (ROI), both for computational efficiency and for interpretability, as will be described later.

At the start of the aggregation process, each searchlight is by itself and has an associated binary information vector with 66 entries corresponding to which pairs of classes can be distinguished in its surrounding searchlight (part 3 of Figure 1). For each searchlight we compute the similarity of its information vector with those of all its neighbours, which yields a 3D grid similarity graph. We then take the portion of the graph corresponding to each ROI in the AAL brain atlas [19], and use modularity [1] to divide it into a number of clusters of adjacent searchlights supporting similar distinctions, as shown in panel 5 of Figure 1. After this is done for all ROIs we obtain a partition of the brain into a few hundred clusters, the data-driven searchlights. Figure 2 depicts the granularity of a typical clustering across multiple brain slices of one of the participants.

The similarity measure between two vectors $\mathbf{v}_i$ and $\mathbf{v}_j$ is obtained by computing the number of 1-entries present in both vectors, $\sum_{pairs} \text{AND}(\mathbf{v}_i, \mathbf{v}_j)$, the number of 1-entries present in only one of them, $\sum_{pairs} \text{XOR}(\mathbf{v}_i, \mathbf{v}_j)$ and then the measure

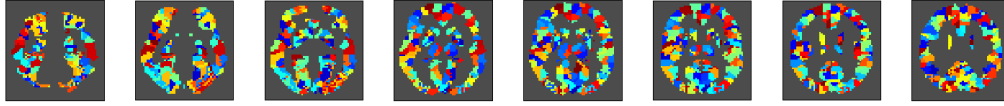

Figure 2: Data-driven searchlights for participant P1 (brain slices range from inferior to superior).

$$similarity(\mathbf{v}_i, \mathbf{v}_j) = \frac{\sum_{pairs} \text{AND}(\mathbf{v}_i, \mathbf{v}_j) - \frac{\sum_{pairs} \text{XOR}(\mathbf{v}_i, \mathbf{v}_j)}{2}}{\sum_{pairs} \text{AND}(\mathbf{v}_i, \mathbf{v}_j)}$$

The measure was chosen because it peaks at 1, if the two vectors match exactly, and decreases – possibly into negative values – if there are mismatches; it will tolerate more mismatches if there are more distinctions being made. It will also deem sparse vectors similar as long as there are vew few mismatches. The number of entries present in only one is divided by 2 so that the differences do not get twice the weight of the similarities.

The centroid for each cluster encodes the pairs of categories that can be distinguished in that data-driven searchlight. The centroid is obtained by combining the binary information vectors for each of the searchlights in it using a soft-AND function, and is itself a binary information vector. A given entry is 1 – the respective pair of categories is distinguishable – if it is 1 in at least $q\%$ of the cluster members (where $q$ is the false discovery rate used earlier to threshold the binary image for that pair of categories).

### 2.2.2 Generation of pattern features from each data-driven searchlight

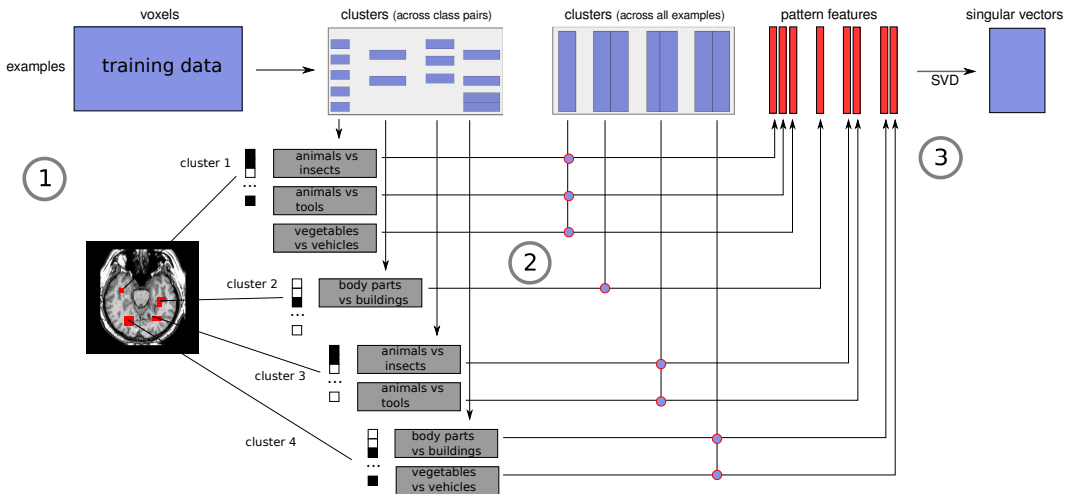

Figure 3: Construction of pattern detectors and pattern features from data-driven searchlights.

**Construct two-way classifiers from each data-driven searchlight**   Each data-driven searchlight has a set of pairs of categories that can be distinguished in it. This indicates that there are particular patterns of activation across the voxels in it which are characteristic of one or more categories, and absent in others. We can leverage this to convert the pattern of activation across the brain into a series of sub-patterns, one from each data-driven searchlight.

For each data-driven searchlight, and for each pairwise category distinction in its information vector, we train a classifier using examples of the two categories and just the voxels in the searchlight (a linear SVM with $\lambda = 1$, [3]); these will be *pattern detectors*, outputting a probability estimate for the prediction (which we transform to the $[-1, 1]$ range), shown in part 1 of Figure 3.

**Use two-way classifiers to generate pattern features** The set of pattern-detectors learned from each data-driven searchlight can be applied to *any* example, not just the ones from the categories that were used to learn them. The output of each pattern-detector is then viewed as representing the degree to which the detector thinks that either of the patterns it is sensitive to is present. For each data-driven searchlight, we apply all of its detectors to *all* the examples in the training set, over the voxels belonging to the searchlight, as illustrated in part 2 of Figure 3. The output of each detector across all examples becomes a new, synthetic *pattern feature*. The number of these pattern features varies per searchlight, as does the number of searchlights per subject, but at the end we will typically have between 10K and 20K of them.

Note that there may be multiple classifiers for a given cluster which produce very similar outputs (e.g. ones that captured a pattern present in all animate object categories versus one present in all inanimate object ones); these will be highly correlated and redundant. We address this by using Singular Value Decomposition (SVD, [7]) to reduce the dimensionality of the matrix of pattern features to the same as the number of examples (180), keeping all singular vectors; this is shown in part 3 of Figure 3. The detectors and the SVD transformation matrix learned from the training set are also applied to the test set.

### 2.2.3 Classification and impact maps for each class

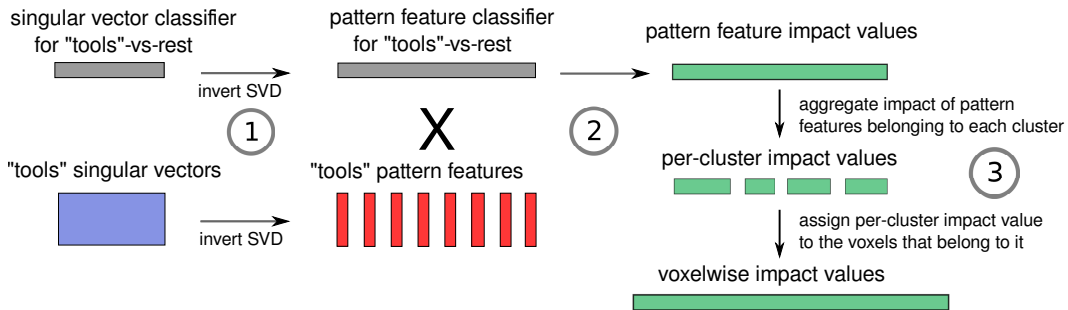

Figure 4: The process of going from the weights of a one-versus-rest category classifier over a low-dimensional pattern feature representation to the impact of each voxel in that classification.

Given the low-dimensional pattern feature dataset, we train a one-versus-rest classifier (a linear SVM with $\lambda = 1$, [3]) for each category; these are then applied to each example in the test set, with the label prediction corresponding to the class with the highest class probability.

The classifiers can also be used to determine the extent to which each data-driven searchlight was responsible for correctly predicting each class. A one-versus-rest category classifier consists of a vector of 180 weights, which can be converted into an equivalent classifier over pattern features by inverting the SVD, as shown in part 1 of Figure 4. The *impact* of each pattern feature in correctly predicting this category can be calculated by multiplying each weight by the values taken by the corresponding pattern feature over examples in the category, and averaging across all examples; this is shown in part 2 of Figure 4. These pattern-feature impact values can then be aggregated by the data-driven searchlight they came from, yielding a net impact value for that searchlight. This is the value that is propagated to each voxel in the data-driven searchlight (part 3 of Figure 4) in order to generate an impact map.

## 3 Experiments and Discussion

### 3.1 Classification

Our goal in this experiment is to determine whether transforming the data from voxel features to pattern features preserves information, and how competitive the results are with a classifier combined with voxel selection. In all experiments we use a split-half cross-validation loop, where the halves contain examples from even and odd epochs, respectively, 180 examples in each (15 per cat-

egory). If cross-validation inside a split-half training set is required, we use leave-one-epoch out cross-validation,

**Baseline** We contrasted experimental results obtained with our method with a baseline of classification using voxel selection. The scoring criterion used to rank each voxel was the accuracy of a LDA classifier – same as described above – using the $3 \times 3 \times 3$ searchlight around each voxel to do 12-category classification. The number of voxels to use was selected by nested cross-validation inside the training set [3]. The classifier used was a linear SVM ($\lambda = 1$, [3]), same as the whole brain classifier in our method.

**Results** The results are shown in the first line of Table 1; across subjects, our method is better than voxel selection, with the $p$-value of a sign-test of this being $< 0.01$. It is substantially better than a classifier using all the voxels in the brain directly.

Whereas the accuracy is above chance ($0.08$) for all subjects, it is rather low for some. There are at least two factors responsible for this. The first is that some classes give rise to very similar patterns of activation (e.g. "buildings" and "building parts"), and hence examples in these classes are confusable (confusion matrices bear this out). The second factor is that subjects vary in their ability to stay focused on the task and avoid stray thoughts or remembering other parts of the experiment, hence examples may not belong to the class corresponding to the label or even any class at all. [13] also points out that accuracy is correlated with a subject's ability to stay still during the experiment.

Table 1: Classification accuracy for the 9 subjects using our method, as well as two baselines.

|  | P1 | P2 | P3 | P4 | P5 | P6 | P7 | P8 | P9 |
|---|---|---|---|---|---|---|---|---|---|
| **our method** | 0.54 | 0.34 | 0.33 | 0.42 | 0.15 | 0.19 | 0.22 | 0.21 | 0.16 |
| baseline (voxel selection) | 0.53 | 0.33 | 0.24 | 0.34 | 0.14 | 0.16 | 0.21 | 0.20 | 0.15 |
| baseline (using all voxels) | 0.31 | 0.21 | 0.19 | 0.27 | 0.13 | 0.09 | 0.14 | 0.13 | 0.15 |
| #voxels selected (fold 1) | 1200 | 400 | 200 | 1600 | 800 | 800 | 800 | 400 | 2000 |
| #voxels selected (fold 2) | 800 | 200 | 100 | 800 | 50 | 8000 | 100 | 1200 | 100 |

## 3.2 Impact maps

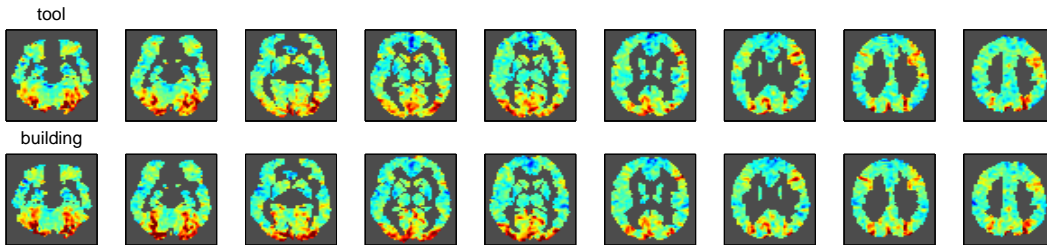

Figure 5: Average example for categories "tool" and "building" in participant P1 (slices ordered from inferior to superior, red is activation above the image mean, blue below).

As described in Section 2.2.3, an impact map can be produced for each category, showing the extent to which each data-driven searchlight helped classify that category correctly. In order to better understand better how impact works, consider two categories "tools" and "buildings" where we know where some of the information resides (for "tools" around the central sulcus, visible on the right of slices to the right, for "buildings" around the parahippocampal gyrus, visible on the lower side of slices to the left). Figure 5 shows the average example for the two categories; note how similar the two examples are across the slices, indicating that most activation is shared between the two categories.

The impact maps for the same participant in Figure 6 show that much of the common activation is eliminated, and that the areas known to be informative are assigned high impact in their respective

tool

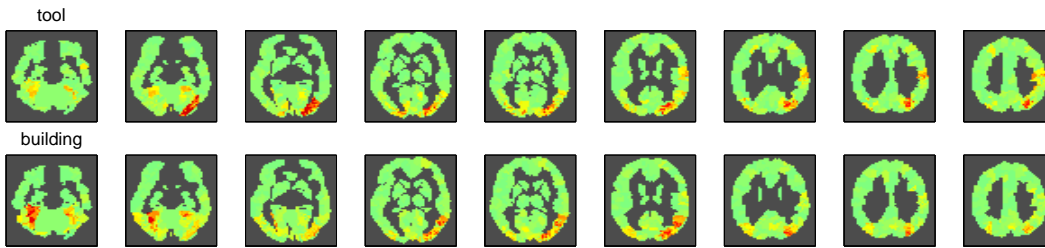

building

Figure 6: Impact map for categories "tool" and "building" in participant P1.

tool

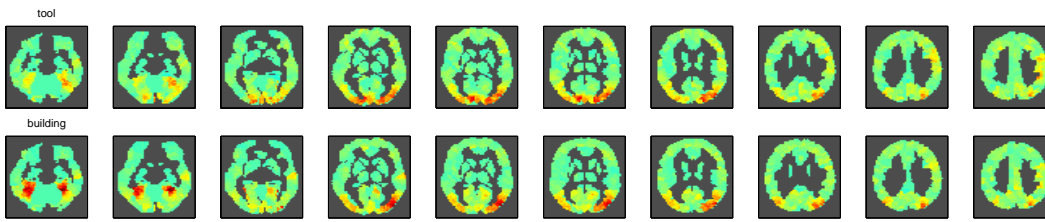

building

Figure 7: Average impact map for categories "tool" and "building" across the nine participants.

maps. Impact is positive, regardless of whether activation in each voxel involved is above or below the mean of the image; the activation of each voxel influences the classifier only in the context of its neighbours in each data-driven searchlight. Note, also, that unlike a simple one-vs-rest classifier or searchlight map, the notion of impact can accommodate the situation where the same location is useful, with either different or the same pattern of activation, for two separate classes (rather than have it be downweighted relative to others that might be unique to that particular class).

Finally, consider that impact maps can be averaged across subjects, as shown in Figure 7, or undergo $t$-tests or a more complex second-level group analysis. A more exploratory analysis can be performed by considering locations that are high impact for every participant and, through their data-driven searchlight, examine the corresponding cluster centroids and get a complete picture of which subsets of the classes can be distinguished there (similar to the bottom-up process in part 5 of Figure 1, but now done top-down and given a cross-validated classification result and impact value).

## Footnotes

[1]Interpretation is more complicated if nonlinear classifiers are being used [6], [17], but this is far less common

[2]The data were kindly shared with us by Tom Mitchell and Marcel Just, from Carnegie Mellon University.

[3]Possible choices were 50, 100, 200, 400, 800, 1200, 1600, 2000, 4000, 8000, 16000 or all voxels.

## References

[1] VD Blondel, JL Guillaume, R Lambiotte, and E Lefebvre. Fast unfolding of communities in large networks. *Journal of Statistical Mechanics: Theory and Experiment*, (10):1–12, 2008.

[2] Melissa K Carroll, Guillermo a Cecchi, Irina Rish, Rahul Garg, and a Ravishankar Rao. Prediction and interpretation of distributed neural activity with sparse models. *NeuroImage*, 44(1):112–22, January 2009.

[3] C.C. Chang and C.J. Lin. LIBSVM: a library for support vector machines. Technical report, 2001.

[4] Karl J Friston, John Ashburner, Stefan J Kiebel, Thomas E Nichols, and W D Penny. *Statistical Parametric Mapping: The Analysis of Functional Brain Images*. Academic Press, 2006.

[5] Christopher R Genovese, Nicole a Lazar, and Thomas Nichols. Thresholding of statistical maps in functional neuroimaging using the false discovery rate. *NeuroImage*, 15(4):870–8, 2002.

[6] Stephen José Hanson, Toshihiko Matsuka, and James V Haxby. Combinatorial codes in ventral temporal lobe for object recognition: Haxby (2001) revisited: is there a "face" area? *NeuroImage*, 23(1):156–66, 2004.

[7] Trevor Hastie, Robert Tibshirani, and Jerome Friedman. *The elements of statistical learning: data mining, inference and prediction*. Springer-Verlag, 2001.

[8] J. Haynes and G. Rees. Decoding mental states from brain activity in humans. *Nature Reviews Neuroscience*, 7(7):523–34, 2006.

[9] Marcel Adam Just, Vladimir L Cherkassky, Sandesh Aryal, and Tom M Mitchell. A neurosemantic theory of concrete noun representation based on the underlying brain codes. *PloS one*, 5(1):e8622, 2010.

[10] N Kriegeskorte, R. Goebel, and P. Bandettini. Information-based functional brain mapping. *Proceedings of the National Academy of Sciences*, 103(10):3863, 2006.

[11] John Langford. Tutorial on Practical Prediction Theory for Classification. *Journal of Machine Learning Research*, 6:273–306, 2005.

[12] T. M. Mitchell, R. Hutchinson, R. S. Niculescu, F. Pereira, X. Wang, M. Just, and S. Newman. Learning to Decode Cognitive States from Brain Images. *Machine Learning*, 57(1/2):145–175, October 2004.

[13] T. M. Mitchell, S. V. Shinkareva, A. Carlson, K. Chang, V. L. Malave, R. A. Mason, and M. A. Just. Predicting human brain activity associated with the meanings of nouns. *Science*, 320(5880):1191–5, 2008.

[14] K. A. Norman, S. M. Polyn, G. J. Detre, and J. V. Haxby. Beyond mind-reading: multi-voxel pattern analysis of fMRI data. *Trends in cognitive sciences*, 10(9):424–30, 2006.

[15] F Pereira and M Botvinick. Classification of functional magnetic resonance imaging data using informative pattern features. *Proceedings of the 17th ACM SIGKDD international conference on Knowledge discovery and data mining - KDD '11*, page 940, 2011.

[16] F. Pereira and M. Botvinick. Information mapping with pattern classifiers: a comparative study. *NeuroImage*, 56(2):835–850, 2011.

[17] Peter Mondrup Rasmussen, Kristoffer Hougaard Madsen, Torben Ellegaard Lund, and Lars Kai Hansen. Visualization of nonlinear kernel models in neuroimaging by sensitivity maps. *NeuroImage*, 55(3):1120–31, April 2011.

[18] Juliane Schäfer and Korbinian Strimmer. A shrinkage approach to large-scale covariance matrix estimation and implications for functional genomics. *Statistical applications in genetics and molecular biology*, 4:Article32, January 2005.

[19] N Tzourio-Mazoyer, B Landeau, D Papathanassiou, F Crivello, O Etard, N Delcroix, B Mazoyer, and M Joliot. Automated anatomical labeling of activations in SPM using a macroscopic anatomical parcellation of the MNI MRI single-subject brain. *NeuroImage*, 15(1):273–89, 2002.

